# Conditional Random Fields with High-Order Features for Sequence Labeling

**Nan Ye**    **Wee Sun Lee**
Department of Computer Science
National University of Singapore
{yenan,leews}@comp.nus.edu.sg

**Hai Leong Chieu**
DSO National Laboratories
chaileon@dso.org.sg

**Dan Wu**
Singapore MIT Alliance
National University of Singapore
dwu@nus.edu.sg

## Abstract

Dependencies among neighbouring labels in a sequence is an important source of information for sequence labeling problems. However, only dependencies between adjacent labels are commonly exploited in practice because of the high computational complexity of typical inference algorithms when longer distance dependencies are taken into account. In this paper, we show that it is possible to design efficient inference algorithms for a conditional random field using features that depend on long consecutive label sequences (high-order features), as long as the number of distinct label sequences used in the features is small. This leads to efficient learning algorithms for these conditional random fields. We show experimentally that exploiting dependencies using high-order features can lead to substantial performance improvements for some problems and discuss conditions under which high-order features can be effective.

## 1    Introduction

In a sequence labeling problem, we are given an input sequence $\mathbf{x}$ and need to label each component of $\mathbf{x}$ with its class to produce a label sequence $\mathbf{y}$. Examples of sequence labeling problems include labeling words in sentences with its type in named-entity recognition problems [16], handwriting recognition problems [15], and deciding whether each DNA base in a DNA sequence is part of a gene in gene prediction problems [2].

Conditional random fields (CRF) [8] has been successfully applied in many sequence labeling problems. Its chief advantage lies in the fact that it models the conditional distribution $P(\mathbf{y}|\mathbf{x})$ rather than the joint distribution $P(\mathbf{y}, \mathbf{x})$. In addition, it can effectively encode arbitrary dependencies of $\mathbf{y}$ on $\mathbf{x}$ as the learning cost mainly depends on the parts of $\mathbf{y}$ involved in the dependencies. However, the use of high-order features, where a feature of order $k$ is a feature that encodes the dependency between $\mathbf{x}$ and $(k+1)$ consecutive elements in $\mathbf{y}$, can potentially lead to an exponential blowup in the computational complexity of inference. Hence, dependencies are usually assumed to exist only between adjacent components of $\mathbf{y}$, giving rise to linear-chain CRFs which limits the order of the features to one.

In this paper, we show that it is possible to learn and predict CRFs with high-order features efficiently under the following *pattern sparsity assumption* (which is often observed in real problems): the number of observed label sequences of length, say $k$, that the features depend on, is much smaller than $n^k$, where $n$ is the number of possible labels. We give an algorithm for computing the marginals and the CRF log likelihood gradient that runs in time polynomial in the number and length of the label sequences that the features depend on. The gradient can be used with quasi-newton methods to efficiently solve the convex log likelihood optimization problem [14]. We also provide an efficient decoding algorithm for finding the most probable label sequence in the presence of long label sequence features. This can be used with cutting plane methods to train max-margin solutions for sequence labeling problems in polynomial time [18].

We show experimentally that using high-order features can improve performance in sequence labeling problems. We show that in handwriting recognition, using even simple high-order indicator features improves performance over using linear-chain CRFs, and impressive performance improvement is observed when the maximum order of the indicator features is increased. We also use a synthetic data set to discuss the conditions under which higher order features can be helpful. We further show that higher order label features can sometimes be more stable under change of data distribution using a named entity data set.

## 2   Related Work

Conditional random fields [8] are discriminately trained, undirected Markov models, which has been shown to perform well in various sequence labeling problems. Although a CRF can be used to capture arbitrary dependencies among components of $\mathbf{x}$ and $\mathbf{y}$, in practice, this flexibility of the CRF is not fully exploited as inference in Markov models is NP-hard in general (see e.g. [1]), and can only be performed efficiently for special cases such as linear chains. As such, most applications involving CRFs are limited to some *tractable* Markov models. This observation also applies to other structured prediction methods such as structured support vector machines [15, 18].

A commonly used inference algorithm for CRF is the clique tree algorithm [5]. Defining a feature depending on $k$ (not necessarily consecutive) labels will require forming a clique of size $k$, resulting in a clique-tree with tree-width greater or equal to $k$. Inference on such a clique tree will be exponential in $k$. For sequence models, a feature of order $k$ can be incorporated into a $k$-order Markov chain, but the complexity of inference is again exponential in $k$. Under the pattern sparsity assumption, our algorithm achieves efficiency by maintaining only information related to a few occurred patterns, while previous algorithms maintain information about all (exponentially many) possible patterns.

In the special case of a semi-Markov random fields, where high-order features depend on segments of *identical* labels, the complexity of inference is linear in the maximum length of the segments [13]. The semi-Markov assumption can be seen as defining a *sparse* feature representation: though the number of length $k$ label patterns is exponential in $k$, the semi-Markov assumption effectively allows only $n^2$ of them ($n$ is the cardinality of the label set), as the features are defined on a sequence of identical labels that can only depend on the label of the preceding segment. Compared to this approach, our algorithm has the advantage of being able to efficiently handle high-order features having arbitrary label patterns.

Long distance dependencies can also be captured using hierarchical models such as Hierarchical Hidden Markov Model (HHMM) [4] or Probabilistic Context Free Grammar (PCFG) [6]. The time complexity of inference in an HHMM is $O(\min\{nl^3, n^2l\})$ [4, 10], where $n$ is the number of states and $l$ is the length of the sequence. Discriminative versions such as hierarchical CRF has also been studied [17]. Inference in PCFG and its discriminative version can also be efficiently done in $O(ml^3)$ where $m$ is the number of productions in the grammar [6]. These methods are able to capture dependencies of arbitrary lengths, unlike $k$-order Markov chains. However, to do efficient learning with these methods, the hierarchical structure of the examples need to be provided. For example, if we use PCFG to do named entity recognition, we need to provide the parse trees for efficient learning; providing the named entity labels for each word is not sufficient. Hence, a training set that has not been labeled with hierarchical labels will need to be relabeled before it can be trained efficiently. Alternatively, methods that employ hidden variables can be used (e.g. to infer the hidden parse tree) but the optimization problem is no longer convex and local optima can sometimes be a problem. Using high-order features captures less expressive form of dependencies than these models but allows efficient learning without relabeling the training set with hierarchical labels.

Similar work on using higher order features for CRFs was independently done in [11]. Their work apply to a larger class of CRFs, including those requiring exponential time for inference, and they did not identify subclasses for which inference is guaranteed to be efficient.

## 3   CRF with High-order Features

Throughout the remainder of this paper, $\mathbf{x}$, $\mathbf{y}$, $\mathbf{z}$ (with or without decorations) respectively denote an observation sequence of length $T$, a label sequence of length $T$, and an arbitrary label sequence. The function $|\cdot|$ denotes the length of any sequence. The set of labels is $\mathcal{Y} = \{1, \dots, n\}$. If

$\mathbf{z} = (y_1, \ldots, y_t)$, then $\mathbf{z}_{i:j}$ denotes $(y_i, \ldots, y_j)$. When $j < i$, $\mathbf{z}_{i:j}$ is the empty sequence (denoted by $\epsilon$). Let the features being considered be $f_1, \ldots, f_m$. Each feature $f_i$ is associated with a label sequence $\mathbf{z}^i$, called $f_i$'s *label pattern*, and $f_i$ has the form

$$f_i(\mathbf{x}, \mathbf{y}, t) = \begin{cases} g_i(\mathbf{x}, t), & \text{if } \mathbf{y}_{t-|\mathbf{z}^i|+1:t} = \mathbf{z}^i \\ 0, & \text{otherwise.} \end{cases}$$

We call $f_i$ a feature of *order* $|\mathbf{z}^i| - 1$. Consider, for example, the problem of named entity recognition. The observations $\mathbf{x} = (x_1, \ldots, x_T)$ may be a word sequence; $g_i(\mathbf{x}, t)$ may be an indicator function for whether $x_t$ is capitalized or may output a precomputed term weight if $x_t$ matches a particular word; and $\mathbf{z}^i$ may be a sequence of two labels, such as (*person*, *organization*) for the named entity recognition task, giving a feature of order one.

A CRF defines conditional probability distributions $P(\mathbf{y}|\mathbf{x}) = Z_{\mathbf{x}}(\mathbf{y})/Z_{\mathbf{x}}$, where $Z_{\mathbf{x}}(\mathbf{y}) = \exp(\sum_{i=1}^m \sum_{t=|\mathbf{z}^i|}^T \lambda_i f_i(\mathbf{x}, \mathbf{y}, t))$, and $Z_{\mathbf{x}} = \sum_{\mathbf{y}} Z_{\mathbf{x}}(\mathbf{y})$. The normalization factor $Z_{\mathbf{x}}$ is called the *partition function*. In this paper, we will use the notation $\sum_{x:Pred(x)} f(x)$ to denote the summation of $f(x)$ over all elements of $x$ satisfying the predicate $Pred(x)$.

## 3.1 Inference for High-order CRF

In this section, we describe the algorithms for computing the partition function, the marginals and the most likely label sequence for high-order CRFs. We give rough polynomial time complexity bounds to give an idea of the effectiveness of the algorithms. These bounds are pessimistic compared to practical performance of the algorithms. It can also be verified that the algorithms for linear chain CRF [8] are special cases of our algorithms when only zero-th and first order features are considered. We show a work example illustrating the computations in the supplementary material.

### 3.1.1 Basic Notations

As in the case of hidden Markov models (HMM) [12], our algorithm uses a forward and backward pass. First, we describe the equivalent of states used in the forward and backward computation. We shall work with three sets: the *pattern set* $\mathcal{Z}$, the *forward-state set* $\mathcal{P}$ and the *backward-state set* $\mathcal{S}$. The pattern set, $\mathcal{Z}$, is the set of distinct label patterns used in the $m$ features. For notational simplicity, assume $\mathcal{Z} = \{\mathbf{z}^1, \ldots, \mathbf{z}^M\}$. The forward-state set, $\mathcal{P} = \{\mathbf{p}^1, \ldots \mathbf{p}^{|\mathcal{P}|}\}$, consists of distinct elements in $\mathcal{Y} \cup \{\mathbf{z}^j_{1:k}\}_{0 \le k \le |\mathbf{z}^j|-1, 1 \le j \le M}$; that is, $\mathcal{P}$ consists of all labels and all proper prefixes (including $\epsilon$) of label patterns, with duplicates removed. Similarly, $\mathcal{S} = \{\mathbf{s}^1, \ldots \mathbf{s}^{|\mathcal{S}|}\}$ consists of the labels and proper suffixes: distinct elements in $\mathcal{Y} \cup \{\mathbf{z}^j_{1:k}\}_{1 \le k \le |\mathbf{z}^j|, 1 \le j \le M}$.

The transitions between states are based on the prefix and suffix relationships defined below. Let $\mathbf{z}_1 \le^p \mathbf{z}_2$ denote that $\mathbf{z}_1$ is a prefix of $\mathbf{z}_2$ and let $\mathbf{z}_1 \le^s \mathbf{z}_2$ denote that $\mathbf{z}_1$ is a suffix of $\mathbf{z}_2$. We define the *longest* prefix and suffix relations with respect to the sets $\mathcal{P}$ and $\mathcal{S}$ as follows

$$\begin{array}{lll} \mathbf{z}_1 \le^p_{\mathcal{S}} \mathbf{z}_2 & \text{if and only if} & (\mathbf{z}_1 \in \mathcal{S}) \text{ and } (\mathbf{z}_1 \le^p \mathbf{z}_2) \text{ and } (\forall \mathbf{z} \in \mathcal{S}, \mathbf{z} \le^p \mathbf{z}_2 \Rightarrow \mathbf{z} \le^p \mathbf{z}_1) \\ \mathbf{z}_1 \le^s_{\mathcal{P}} \mathbf{z}_2 & \text{if and only if} & (\mathbf{z}_1 \in \mathcal{P}) \text{ and } (\mathbf{z}_1 \le^s \mathbf{z}_2) \text{ and } (\forall \mathbf{z} \in \mathcal{P}, \mathbf{z} \le^s \mathbf{z}_2 \Rightarrow \mathbf{z} \le^s \mathbf{z}_1). \end{array}$$

Finally, the subsequence relationship defined below are used when combining forward and backward variables to compute marginals. Let $\mathbf{z} \subseteq \mathbf{z}'$ denote that $\mathbf{z}$ is a subsequence of $\mathbf{z}'$, $\mathbf{z} \subset \mathbf{z}'$ denote that $\mathbf{z}$ is a subsequence of $\mathbf{z}'_{2:|\mathbf{z}'|-1}$. The addition of subscript $j$ in $\subseteq_j$ and $\subset_j$ indicates that the condition $\mathbf{z} \le^s \mathbf{z}'_{1:j}$ is satisfied as well (that is, $\mathbf{z}$ ends at position $j$ in $\mathbf{z}'$).

We shall give rough time bounds in terms of $m$ (the total number of features), $n$ (the number of labels), $T$ (the length of the sequence), $M$ (the number of distinct label patterns in $\mathcal{Z}$), and the maximum order $K = \max\{|\mathbf{z}^1| - 1, \ldots, |\mathbf{z}^M| - 1\}$.

### 3.1.2 The Forward and Backward Variables

We now define forward vector $\alpha_{\mathbf{x}}$ and backward vector $\beta_{\mathbf{x}}$. Suppose $\mathbf{z} \le^p \mathbf{y}$, then define $\mathbf{y}$'s prefix score $Z^p_{\mathbf{x}}(\mathbf{z}) = \exp(\sum_{i=1}^m \sum_{t=|\mathbf{z}^i|}^{|\mathbf{z}|} \lambda_i f_i(\mathbf{x}, \mathbf{y}, t))$. Similarly, if $\mathbf{z} \le^s \mathbf{y}$, then define $\mathbf{y}$'s suffix score

$Z_{\mathbf{x}}^s(\mathbf{z}) = \exp(\sum_{i=1}^m \sum_{t=T-|\mathbf{z}|+|\mathbf{z}^i|}^T \lambda_i f_i(\mathbf{x}, \mathbf{y}, t))$. $Z_{\mathbf{x}}^p(\mathbf{z})$ and $Z_{\mathbf{x}}^s(\mathbf{z})$ only depend on $\mathbf{z}$. Let

$$\alpha_{\mathbf{x}}(t, \mathbf{p}^i) = \sum_{\mathbf{z}:|\mathbf{z}|=t, \mathbf{p}^i \leq_{\mathcal{P}}^s \mathbf{z}} Z_{\mathbf{x}}^p(\mathbf{z})$$

$$\beta_{\mathbf{x}}(t, \mathbf{s}^i) = \sum_{\mathbf{z}:|\mathbf{z}|=T+1-t, \mathbf{s}^i \leq_{\mathcal{S}}^p \mathbf{z}} Z_{\mathbf{x}}^s(\mathbf{z}).$$

The variable $\alpha_{\mathbf{x}}(t, \mathbf{p}^i)$ computes for $\mathbf{x}_{1:t}$ the sum of the scores of all its label sequences $\mathbf{z}$ having $\mathbf{p}^i$ as the longest suffix. Similarly, the variable $\beta_{\mathbf{x}}(t, \mathbf{s}^i)$ computes for $\mathbf{x}_{t:T}$ the sum of scores of all its label sequence $\mathbf{z}$ having $\mathbf{s}^i$ as the longest prefix. Each vector $\alpha_{\mathbf{x}}(t, \cdot)$ is of dimension $|\mathcal{P}|$, while $\beta_{\mathbf{x}}(t, \cdot)$ has dimension $|\mathcal{S}|$. We shall compute the $\alpha_{\mathbf{x}}$ and $\beta_{\mathbf{x}}$ vectors with dynamic programming.

Let $\Psi_{\mathbf{x}}^p(t, \mathbf{p}) = \exp(\sum_{i:\mathbf{z}^i \leq^s \mathbf{p}} \lambda_i g_i(\mathbf{x}, t))$. For $\mathbf{y}$ with $\mathbf{p} \leq^s \mathbf{y}_{1:t}$, this function counts the contribution towards $Z_{\mathbf{x}}(\mathbf{y})$ by all features $f_i$ with their label patterns ending at position $t$ and being suffixes of $\mathbf{p}$. Let $\mathbf{p}^i y$ be the concatenation of $\mathbf{p}^i$ with a label $y$. The following proposition is immediate.

**Proposition 1**      (a) For any $\mathbf{z}$, there is a unique $\mathbf{p}^i$ such that $\mathbf{p}^i \leq_{\mathcal{P}}^s \mathbf{z}$.

    (b) For any $\mathbf{z}$, $y$, if $\mathbf{p}^i \leq_{\mathcal{P}}^s \mathbf{z}$ and $\mathbf{p}^k \leq_{\mathcal{P}}^s \mathbf{p}^i y$, then $\mathbf{p}^k \leq_{\mathcal{P}}^s \mathbf{z}y$ and $Z_{\mathbf{x}}^p(\mathbf{z}y) = \Psi_{\mathbf{x}}^p(t, \mathbf{p}^i y)Z_{\mathbf{x}}^p(\mathbf{z})$.

Proposition 1(a) means that we can induce partitions of label sequences using the forward states. and Proposition 1(b) shows how to make well-defined transition from one forward state at a time slice to another forward state at the next time slice. By definition, $\alpha_{\mathbf{x}}(0, \epsilon) = 1$, and $\alpha_{\mathbf{x}}(0, \mathbf{p}^i) = 0$ for all $\mathbf{p}^i \neq \epsilon$. Using Proposition 1(b), the recurrence for $\alpha_{\mathbf{x}}$ is

$$\alpha_{\mathbf{x}}(t, \mathbf{p}^k) = \sum_{(\mathbf{p}^i, y):\mathbf{p}^k \leq_{\mathcal{P}}^s \mathbf{p}^i y} \Psi_{\mathbf{x}}^p(t, \mathbf{p}^i y)\alpha_{\mathbf{x}}(t-1, \mathbf{p}^i), \text{ for } 1 \leq t \leq T.$$

Similarly, for the backward vectors $\beta_{\mathbf{x}}$, let $\Psi_{\mathbf{x}}^s(t, \mathbf{s}) = \exp(\sum_{i:\mathbf{z}^i \leq^p \mathbf{s}} \lambda_i g_i(\mathbf{x}, t + |\mathbf{z}^i| - 1))$. By definition, $\beta_{\mathbf{x}}(T+1, \epsilon) = 1$, and $\beta_{\mathbf{x}}(T+1, \mathbf{s}^i) = 0$ for all $\mathbf{s}^i \neq \epsilon$. The recurrence for $\beta_{\mathbf{x}}$ is

$$\beta_{\mathbf{x}}(t, \mathbf{s}^k) = \sum_{(\mathbf{s}^i, y):\mathbf{s}^k \leq_{\mathcal{S}}^p y\mathbf{s}^i} \Psi_{\mathbf{x}}^s(t, y\mathbf{s}^i)\beta_{\mathbf{x}}(t+1, \mathbf{s}^i), \text{ for } 1 \leq t \leq T.$$

Once $\alpha_{\mathbf{x}}$ or $\beta_{\mathbf{x}}$ is computed, then using Proposition 1(a), $Z_{\mathbf{x}}$ can be easily obtained:

$$Z_{\mathbf{x}} = \sum_{i=1}^{|\mathcal{P}|} \alpha_{\mathbf{x}}(T, \mathbf{p}^i) = \sum_{i=1}^{|\mathcal{S}|} \beta_{\mathbf{x}}(1, \mathbf{s}^i).$$

**Time Complexity:** We assume that each evaluation of the function $g_i(\cdot, \cdot)$ can be performed in unit time for all $i$. All relevant values of $\Psi_{\mathbf{x}}^p$ that are used can hence be computed in $O(mn|\mathcal{P}|T)$ (thus $O(mnMKT)$) time. In practice, this is pessimistic, and the computation can be done more quickly. For all following analyses, we assume that $\Psi_{\mathbf{x}}^p$ has already been computed and stored in an array. Now all values of $\alpha_{\mathbf{x}}$ can be computed in $\Theta(n|\mathcal{P}|T)$, thus $O(nMKT)$ time. Similar bounds for $\Psi_{\mathbf{x}}^s$ and $\beta_{\mathbf{x}}$ hold.

### 3.1.3 Computing the Most Likely Label Sequence

As in the case of HMM [12], Viterbi decoding (calculating the most likely label sequence) is obtained by replacing the sum operator in the forward backward algorithm with the max operator.

Formally, let $\delta_{\mathbf{x}}(t, \mathbf{p}^i) = \max_{\mathbf{z}:|\mathbf{z}|=t, \mathbf{p}^i \leq_{\mathcal{P}}^s \mathbf{z}} Z_{\mathbf{x}}^p(\mathbf{z})$. By definition, $\delta_{\mathbf{x}}(0, \epsilon) = 1$, and $\delta_{\mathbf{x}}(0, \mathbf{p}^i) = 0$ for all $\mathbf{p}^i \neq \epsilon$, and using Proposition 1, we have

$$\delta_{\mathbf{x}}(t, \mathbf{p}^k) = \max_{(\mathbf{p}^i, y):\mathbf{p}^k \leq_{\mathcal{P}}^s \mathbf{p}^i y} \Psi_{\mathbf{x}}^p(t, \mathbf{p}^i y)\delta_{\mathbf{x}}(t-1, \mathbf{p}^i), \text{ for } 1 \leq t \leq T.$$

We use $\Phi_{\mathbf{x}}(t, \mathbf{p}^k)$ to record the pair $(\mathbf{p}^i, y)$ chosen to obtain $\delta_{\mathbf{x}}(t, \mathbf{p}^k)$,

$$\Phi_{\mathbf{x}}(t, \mathbf{p}^k) = \arg\max_{(\mathbf{p}^i, y):\mathbf{p}^k \leq_{\mathcal{P}}^s \mathbf{p}^i y} \Psi_{\mathbf{x}}^p(t, \mathbf{p}^i y)\delta_{\mathbf{x}}(t-1, \mathbf{p}^i).$$

Let $\mathbf{p}_T^* = \arg\max_{\mathbf{p}^i} \delta_{\mathbf{x}}(T, \mathbf{p}^i)$, then the most likely path $\mathbf{y}^* = (y_1^*, \ldots, y_T^*)$ has $y_T^*$ as the last label in $\mathbf{p}_T^*$, and the full sequence can be traced backwards using $\Phi_{\mathbf{x}}(\cdot, \cdot)$ as follows

$$(\mathbf{p}_t^*, y_t^*) = \Phi_{\mathbf{x}}(t+1, \mathbf{p}_{t+1}^*), \text{ for } 1 \le t < T.$$

**Time Complexity:** Either $\Psi_{\mathbf{x}}^p$ or $\Psi_{\mathbf{x}}^s$ can be used for decoding; hence decoding can be done in $\Theta(n \min\{|\mathcal{P}|, |\mathcal{S}|\} T)$ time.

### 3.1.4 Computing the Marginals

We need to compute marginals of label sequences and single variables, that is, compute $P(\mathbf{y}_{t-|\mathbf{z}|:t} = \mathbf{z}|\mathbf{x})$ for $\mathbf{z} \in \mathcal{Z} \cup \mathcal{Y}$. Unlike in the traditional HMM, additional care need to be taken regarding features having label patterns that are super or sub sequences of $\mathbf{z}$. We define

$$W_{\mathbf{x}}(t, \mathbf{z}) = \exp\Big( \sum_{(i,j):\mathbf{z}^i \subset_j \mathbf{z}} \lambda_i g_i(\mathbf{x}, t - |\mathbf{z}| + j) \Big).$$

This function computes the sum of all features that may activate strictly within $\mathbf{z}$.

If $\mathbf{z}_{1:|\mathbf{z}|-1} \le^s \mathbf{p}^i$ and $\mathbf{z}_{2:|\mathbf{z}|} \le^p \mathbf{s}^j$, define $[\mathbf{p}^i, \mathbf{z}, \mathbf{s}^j]$ as the sequence $\mathbf{p}_{1:|\mathbf{p}^i|-(|\mathbf{z}|-1)}^i \mathbf{z} \mathbf{s}_{|\mathbf{z}|-1:|\mathbf{s}^j|}^j$, and

$$O_{\mathbf{x}}(t, \mathbf{p}^i, \mathbf{s}^j, \mathbf{z}) = \exp\Big( \sum_{(k,k'):\mathbf{z} \subseteq \mathbf{z}^k, \mathbf{z}^k \subseteq_{k'} [\mathbf{p}^i, \mathbf{z}, \mathbf{s}^j]} \lambda_k g_k(\mathbf{x}, t - |\mathbf{p}^i| + k' - 1) \Big).$$

$O_{\mathbf{x}}(t, \mathbf{p}^i, \mathbf{s}^j, \mathbf{z})$ counts the contribution of features with their label patterns properly containing $\mathbf{z}$ but within $[\mathbf{p}^i, \mathbf{z}, \mathbf{s}^j]$.

**Proposition 2** *Let $\mathbf{z} \in \mathcal{Z} \cup \mathcal{Y}$. For any $\mathbf{y}$ with $\mathbf{y}_{t-|\mathbf{z}|+1:t} = \mathbf{z}$, there exists unique $\mathbf{p}^i, \mathbf{s}^j$ such that $\mathbf{z}_{1:|\mathbf{z}|-1} \le^s \mathbf{p}^i$, $\mathbf{z}_{2:|\mathbf{z}|} \le^p \mathbf{s}^j$, $\mathbf{p}^i \le_{\mathcal{P}}^s \mathbf{y}_{1:t-1}$, and $\mathbf{s}^j \le_{\mathcal{S}}^p \mathbf{y}_{t-|\mathbf{z}|+2:T}$. In addition, $Z_{\mathbf{x}}(\mathbf{y}) = \frac{1}{W_{\mathbf{x}}(t,\mathbf{z})} Z_{\mathbf{x}}^p(t-1, \mathbf{y}_{1:t-1}) Z_{\mathbf{x}}^s(T+1-(t-|\mathbf{z}|+2), \mathbf{y}_{t-|\mathbf{z}|+2:T}) O_{\mathbf{x}}(t, \mathbf{p}^i, \mathbf{s}^j, \mathbf{z})$.*

Multiplying by $O_{\mathbf{x}}$ counts features that are not counted in $Z_{\mathbf{x}}^p Z_{\mathbf{x}}^s$ while division by $W_{\mathbf{x}}$ removes features that are double-counted. By Proposition 2, we have

$$P(\mathbf{y}_{t-|\mathbf{z}|+1:t} = \mathbf{z}|\mathbf{x}) = \frac{\sum_{(i,j):\mathbf{z}_{1:|\mathbf{z}|-1} \le^s \mathbf{p}^i, \mathbf{z}_{2:|\mathbf{z}|} \le^p \mathbf{s}^j} \alpha_{\mathbf{x}}(t-1, \mathbf{p}^i) \beta_{\mathbf{x}}(t-|\mathbf{z}|+2, \mathbf{s}^j) O_{\mathbf{x}}(t, \mathbf{p}^i, \mathbf{s}^j, \mathbf{z})}{Z_{\mathbf{x}} W_{\mathbf{x}}(t, \mathbf{z})}.$$

**Time Complexity:** Both $W_{\mathbf{x}}(t, \mathbf{z})$ and $O_{\mathbf{x}}(t, \mathbf{p}^i, \mathbf{s}^j, \mathbf{z})$ can be computed in $O(|\mathbf{p}^i||\mathbf{s}^j|) = O(K^2)$ time (with some precomputation). Thus a very pessimistic time bound for computing $P(\mathbf{y}_{t-|\mathbf{z}|+1:t} = \mathbf{z}|\mathbf{x})$ is $O(K^2|\mathcal{P}||\mathcal{S}|) = O(M^2 K^4)$.

### 3.2 Training

Given a training set $\mathcal{T}$, the model parameters $\lambda_i$'s can be chosen by maximizing the regularized log-likelihood $\mathcal{L}_{\mathcal{T}} = \log \Pi_{(\mathbf{x},\mathbf{y}) \in \mathcal{T}} P(\mathbf{y}|\mathbf{x}) - \sum_{i=1}^m \frac{\lambda_i^2}{2\sigma_{reg}^2}$, where $\sigma_{reg}$ is a parameter that controls the degree of regularization. Note that $\mathcal{L}_{\mathcal{T}}$ is a concave function of $\lambda_1, \ldots, \lambda_m$, and its maximum is achieved when

$$\frac{\partial \mathcal{L}_{\mathcal{T}}}{\partial \lambda_i} = \tilde{E}(f_i) - E(f_i) - \frac{\lambda_k}{\sigma_{reg}^2} = 0$$

where $\tilde{E}(f_i) = \sum_{(\mathbf{x},\mathbf{y}) \in \mathcal{T}} \sum_{t=|\mathbf{z}^i|}^{|\mathbf{x}|} f_i(\mathbf{x}, \mathbf{y}, t)$ is the empirical sum of the feature $f_i$ in the observed data, and $E(f_i) = \sum_{(\mathbf{x},\mathbf{y}) \in \mathcal{T}} \sum_{|\mathbf{y}'|=|\mathbf{x}|} P(\mathbf{y}'|\mathbf{x}) \sum_{t=|\mathbf{z}^i|}^{|\mathbf{x}|} f_i(\mathbf{x}, \mathbf{y}', t)$ is the expected sum of $f_i$. Given the gradient and value of $\mathcal{L}_{\mathcal{T}}$, we use the L-BFGS optimization method [14] for maximizing the regularized log-likelihood.

The function $\mathcal{L}_{\mathcal{T}}$ can now be computed because we have shown how to compute $Z_{\mathbf{x}}$, and computing the value of $Z_{\mathbf{x}}(\mathbf{y})$ is straightforward, for all $(\mathbf{x}, \mathbf{y}) \in \mathcal{T}$. For the gradient, computing $\tilde{E}(f_i)$ is

straightforward, and $E(f_i)$ can be computed using marginals computed in previous section:

$$E(f_i) \quad = \quad \sum_{(\mathbf{x},\mathbf{y})\in\mathcal{T}} \sum_{t=|\mathbf{z}^i|}^{|\mathbf{x}|} P(\mathbf{y}'_{t-|\mathbf{z}^i|+1:t} = \mathbf{z}^i|\mathbf{x})g_i(\mathbf{x},t).$$

**Time Complexity:** Computing the gradient is clearly more time-consuming than $\mathcal{L}_{\mathcal{T}}$, thus we shall just consider the time needed to compute the gradient. Let $X = \sum_{(\mathbf{x},\mathbf{y})\in\mathcal{T}} |\mathbf{x}|$. We need to compute at most $MX$ marginals, thus total time needed to compute all the marginals has $O(M^3K^4X)$ as an upper bound. Given the marginals, we can compute the gradient in $O(mX)$ time. If the total number of gradient computations needed in maximization is $I$, then the total running time in training is bounded by $O((M^3K^4 + m)XI)$ (very pessimistic).

# 4 Experiments

The practical feasibility of making use of high-order features based on our algorithm lies in the observation that the pattern sparsity assumption often holds. Our algorithm can be applied to take those high-order features into consideration; high-order features now form a component that one can play with in feature engineering.

Now, the question is whether high-order features are *practically significant*. We first use a synthetic data set to explore conditions under which high-order features can be expected to help. We then use a handwritten character recognition problem to demonstrate that even incorporating simple high-order features can lead to impressive performance improvement on a naturally occurring dataset. Finally, we use a named entity data set to show that for some data sets, higher order label features may be more robust to changes in data distributions than observation features.

## 4.1 Synthetic Data Generated Using $k$-Order Markov Model

We randomly generate an order $k$ Markov model with $n$ states $s_1, \ldots, s_n$ as follows. To increase pattern sparsity, we allow at most $r$ randomly chosen possible next state given the previous $k$ states. This limits the number of possible label sequences in each length $k + 1$ segment from $n^{k+1}$ to $n^k r$. The conditional probabilities of these $r$ next states is generated by randomly selecting a vector from uniform distribution over $[0, 1]^r$ and normalizing them. Each state $s_i$ generates an observation $(a_1, \ldots, a_m)$ such that $a_j$ follows a Gaussian distribution with mean $\mu_{ij}$ and standard deviation $\sigma$. Each $\mu_{i,j}$ is independently randomly generated from the uniform distribution over $[0, 1]$. In the experiments, we use values of $n = 5$, $r = 2$ and $m = 3$.

The standard deviation, $\sigma$, has an important role in determining the characteristics of the data generated by this Markov model. If $\sigma$ is very small as compared to most $\mu_{ij}$'s, then using the observations alone as features is likely to be good enough to obtain a good classifier of the states; the label correlations becomes less important for classification. However, if $\sigma$ is large, then it is difficult to distinguish the states based on the observations alone and the label correlations, particularly those captured by higher order features are likely to be helpful. In short, the standard deviation, $\sigma$, is used to to control how much information the observations reveal about the states.

We use the current, previous and next observations, rather than just the current observation as features, exploiting the conditional probability modeling strength of CRFs. For higher order features, we simply use all indicator features that appeared in the training data up to a maximum order. We considered the case $k = 2$ and $k = 3$, and varied $\sigma$ and the maximum order. The training set and test set each contains 500 sequences of length 20; each sequence was initialized with a random sequence of length $k$ and generated using the randomly generated order $k$ Markov model. Training was done by maximizing the regularized log likelihood with regularization parameter $\sigma_{\text{reg}} = 1$ in all experiments in this paper. The experimental results are shown in Figure 1.

Figure 1 shows that the high-order indicator features are useful in this case. In particular, we can see that it is beneficial to increase the order of the high-order features when the underlying model has longer distance correlations. As expected, increasing the order of the features beyond the order of the underlying model is not helpful. The results also suggests that in general, if the observations are closely coupled with the states (in the sense that different states correspond to very different observations), then feature engineering on the observations is generally enough to perform well, and

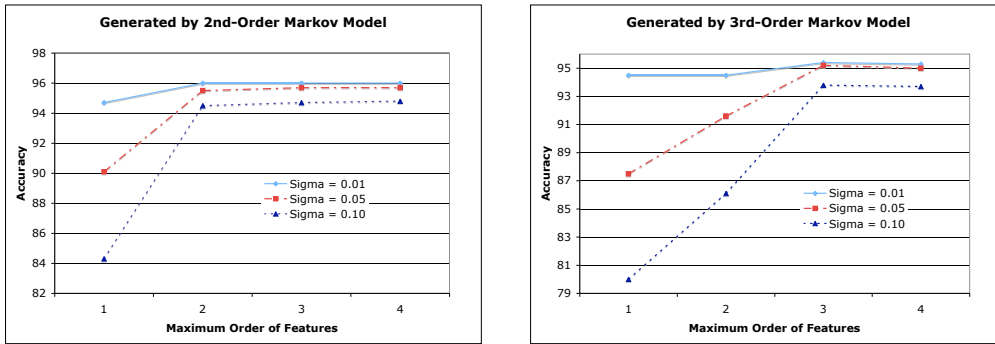

Figure 1: Accuracy as a function of maximum order on the synthetic data set.

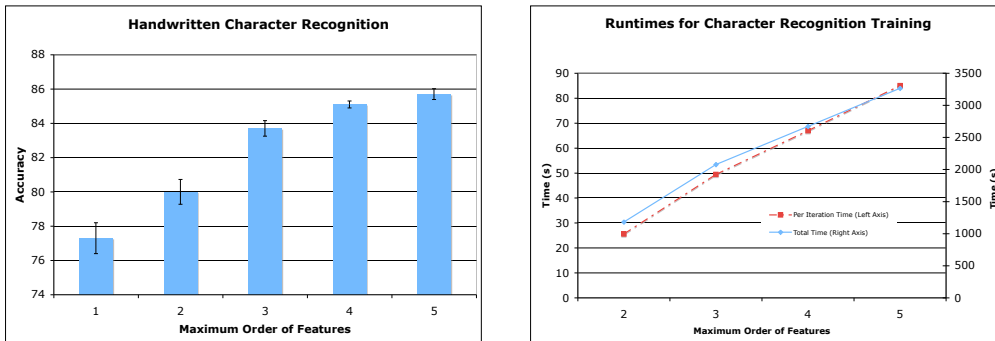

Figure 2: Accuracy (left) and running time (right) as a function of maximum order for the handwriting recognition data set.

it is less important to use high-order features to capture label correlations. On the other hand, when such coupling is not clear, it becomes important to capture the label correlations, and high-order features can be useful.

## 4.2 Handwriting Recognition

We used the handwriting recognition data set from [15], consisting of around 6100 handwritten words with an average length of around 8 characters. The data was originally collected by Kassel [7] from around 150 human subjects. The words were segmented into characters, and each character was converted into an image of 16 by 8 binary pixels. In this labeling problem, each $x_i$ is the image of a character, and each $y_i$ is a lower-case letter. The experimental setup is the same as that used in [15]: the data set was divided into 10 folds with each fold having approximately 600 training and 5500 test examples and the zero-th order features for a character are the pixel values.

For higher order features, we again used all indicator features that appeared in the training data up to a maximum order. The average accuracy over the 10 folds are shown in Figure 2, where strong improvements are observed as the maximum order increases. Figure 2 also shows the total training time and the running time per iteration of the L-BFGS algorithm (which requires computation of the gradient and value of the function at each iteration). The running time appears to grow no more than linearly with the maximum order of the features for this data set.

## 4.3 Named Entity Recognition with Distribution Change

The Named Entity Recognition (NER) problem asks for identification of named entities from texts. With carefully engineered observation features, there does not appear to be very much to be gained from using higher order features. However, in some situations, the training data does not come from the same distribution as the test data. In such cases, we hypothesize that higher order label features may be more stable than observation features and can sometimes offer performance gain.

In our experiment, we used the Automatic Content Extraction (ACE) data [9], which is labeled with seven classes: *Organization*, *Geo-political*, *Location*, *Facility*, *Vehicle*, and *Weapon*. The ACE data

comes from several genres and we use the following in our experiment: Broadcast conversation (BC), Newswire (NW), Weblog (WL) and Usenet (UN).

We use all pairs of genres as training and test data. Scoring was done with the F1 score [16]. The features used are previous word, next word, current word, case patterns for these words, and all indicator label features of order up to $k$. The results for the case $k = 1$ and $k = 2$ are shown in Figure 3. Introducing second order indicator features shows improvement in 10 out of the 12 combinations and degrades performance in two of the combinations. However, the overall effect is small, with an average improvement of 0.62 in F1 score.

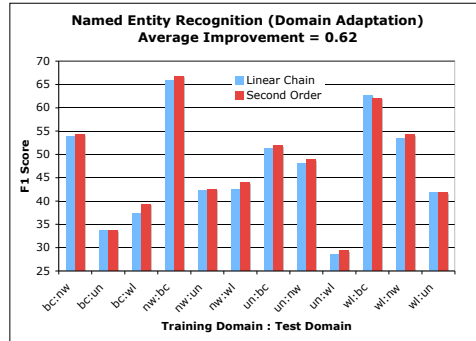

Figure 3: Named entity recognition results.

### 4.4 Discussion

In our experiments, we used indicator features of all label patterns that appear in the training data. For real applications, if the pattern sparsity assumption is not satisfied, but certain patterns do not appear frequently enough and are not really important, then it is useful to see how we can select a subset of features with few distinct label patterns automatically. One possible approach would be to use boosting type methods [3] to sequentially select useful features.

An alternate approach to feature selection is to use all possible features and maximize the margin of the solution instead. Generalization error bounds [15] show that it is possible to obtain good generalization with a relatively small training set size despite having a very large number of features if the margin is large. This indicates that feature selection may not be critical in some cases. Theoretically, it is also interesting to note that minimizing the regularized training cost when all possible high-order features of arbitrary length are used is computationally tractable. This is because the representer theorem [19] tells us that the optimum solution for minimizing quadratically regularized cost functions lies on the span of the training examples. Hence, even when we are learning with arbitrary sets of high-order features, we only need to use the features that appear in the training set to obtain the optimal solution. Given a training set of $N$ sequences of length $l$, only $O(l^2 N)$ long label sequences of all orders are observed. Using cutting plane techniques [18] the computational complexity of optimization is polynomial in inverse accuracy parameter, the training set size and maximum length of the sequences.

It should also be possible to use kernels within the approach here. On the handwritten character problem, [15] reports substantial improvement in performance with the use of kernels. Use of kernels together with high-order features may lead to further improvements. However, we note that the advantage of the higher order features may become less substantial as the observations become more powerful in distinguishing the classes. Whether the use of higher order features together with kernels brings substantial improvement in performance is likely to be problem dependent. Similarly, observation features that are more distribution invariant such as comprehensive name lists can be used for the NER task we experimented with and may reduce the improvements offered by higher order features.

## 5   Conclusion

The pattern sparsity assumption often holds in real applications, and we give efficient inference algorithms for CRF with high-order features when the pattern sparsity assumption is satisfied. This allows high-order features to be explored in feature engineering for real applications. We studied the conditions that are favourable for using high-order features using a synthetic data set, and demonstrated that using simple high-order features can lead to performance improvement on a handwriting recognition problem and a named entity recognition problem.

### Acknowledgements

This work is supported by DSO grant R-252-000-390-592 and AcRF grant R-252-000-327-112.

# References

[1] B. A. Cipra, "The Ising model is NP-complete," *SIAM News*, vol. 33, no. 6, 2000.

[2] A. Culotta, D. Kulp, and A. McCallum, "Gene prediction with conditional random fields," University of Massachusetts, Amherst, Tech. Rep. UM-CS-2005-028, 2005.

[3] T. G. Dietterich, A. Ashenfelter, and Y. Bulatov, "Training conditional random fields via gradient tree boosting," in *Proceedings of the Twenty-First International Conference on Machine Learning*, 2004.

[4] S. Fine, Y. Singer, and N. Tishby, "The hierarchical hidden markov model: Analysis and applications," *Machine Learning*, vol. 32, no. 1, pp. 41–62, 1998.

[5] C. Huang and A. Darwiche, "Inference in belief networks: A procedural guide," *International Journal of Approximate Reasoning*, vol. 15, no. 3, pp. 225–263, 1996.

[6] F. Jelinek, J. D. Lafferty, and R. L. Mercer, "Basic methods of probabilistic context free grammars," in *Speech Recognition and Understanding. Recent Advances, Trends, and Applications*. Springer Verlag, 1992.

[7] R. H. Kassel, "A comparison of approaches to on-line handwritten character recognition," Ph.D. dissertation, Massachusetts Institute of Technology, Cambridge, MA, USA, 1995.

[8] J. Lafferty, A. McCallum, and F. Pereira, "Conditional random fields: Probabilistic models for segmenting and labeling sequence data," in *Proceedings of the Eighteenth International Conference on Machine Learning*, 2001, pp. 282–289.

[9] Linguistic Data Consortium, "ACE (Automatic Content Extraction) English Annotation Guidelines for Entities," 2005.

[10] K. P. Murphy and M. A. Paskin, "Linear-time inference in hierarchical HMMs," in *Advances in Neural Information Processing Systems 14*, vol. 14, 2002.

[11] X. Qian, X. Jiang, Q. Zhang, X. Huang, and L. Wu, "Sparse higher order conditional random fields for improved sequence labeling," in *ICML*, 2009, p. 107.

[12] L. R. Rabiner, *A tutorial on hidden Markov models and selected applications in speech recognition*. San Francisco, CA, USA: Morgan Kaufmann Publishers Inc., 1990.

[13] S. Sarawagi and W. W. Cohen, "Semi-Markov conditional random fields for information extraction," in *Advances in Neural Information Processing Systems 17*. Cambridge, MA: MIT Press, 2005, pp. 1185–1192.

[14] F. Sha and F. Pereira, "Shallow parsing with conditional random fields," in *Proceedings of the Twentieth International Conference on Machine Learning*, 2003, pp. 282–289.

[15] B. Taskar, C. Guestrin, and D. Koller, "Max-margin Markov networks," in *Advances in Neural Information Processing Systems 16*. Cambridge, MA: MIT Press, 2004.

[16] E. Tjong and F. D. Meulder, "Introduction to the CoNLL-2003 shared task: Language-independent named entity recognition," in *Proceedings of Conference on Computational Natural Language Learning*, 2003.

[17] T. T. Tran, D. Phung, H. Bui, and S. Venkatesh, "Hierarchical semi-Markov conditional random fields for recursive sequential data," in *NIPS'08: Advances in Neural Information Processing Systems 20*. Cambridge, MA: MIT Press, 2008, pp. 1657–1664.

[18] I. Tsochantaridis, T. Hofmann, T. Joachims, and Y. Altun, "Support vector machine learning for interdependent and structured output spaces," in *Proceedings of the Twenty-First international conference on Machine learning*, 2004, pp. 104–112.

[19] G. Wahba, *Spline models for observational data*, ser. CBMS-NSF Regional Conference Series in Applied Mathematics. Philadelphia, PA: Society for Industrial and Applied Mathematics (SIAM), 1990, vol. 59.
